# Learning Semantic Similarity

**Jaz Kandola**    **John Shawe-Taylor**
Royal Holloway, University of London
{*jaz, john*}*@cs.rhul.ac.uk*

**Nello Cristianini**
University of California, Berkeley
*nello@support-vector.net*

## Abstract

The standard representation of text documents as bags of words suffers from well known limitations, mostly due to its inability to exploit semantic similarity between terms. Attempts to incorporate some notion of term similarity include latent semantic indexing [8], the use of semantic networks [9], and probabilistic methods [5]. In this paper we propose two methods for inferring such similarity from a corpus. The first one defines word-similarity based on document-similarity and viceversa, giving rise to a system of equations whose equilibrium point we use to obtain a semantic similarity measure. The second method models semantic relations by means of a diffusion process on a graph defined by lexicon and co-occurrence information. Both approaches produce valid kernel functions parametrised by a real number. The paper shows how the alignment measure can be used to successfully perform model selection over this parameter. Combined with the use of support vector machines we obtain positive results.

## 1  Introduction

Kernel-based algorithms exploit the information encoded in the inner-products between all pairs of data items (see for example [1]). This matches very naturally the standard representation used in text retrieval, known as the 'vector space model', where the similarity of two documents is given by the inner product between high dimensional vectors indexed by all the terms present in the corpus. The combination of these two methods, pioneered by [6], and successively explored by several others, produces powerful methods for text categorization. However, such an approach suffers from well known limitations, mostly due to its inability to exploit semantic similarity between terms: documents sharing terms that are different but semantically related will be considered as unrelated. A number of attempts have been made to incorporate semantic knowledge into the vector space representation. Semantic networks have been considered [9], whilst others use co-occurrence analysis where a semantic relation is assumed between terms whose occurrence patterns in the documents of the corpus are correlated [3]. Such methods are also limited in their flexibility, and the question of how to infer semantic relations between terms or documents from a corpus remains an open issue. In this paper we propose two methods to model such relations in an unsupervised way. The structure of the paper is as follows. Section 2 provides an introduction to how semantic similarity can be

introduced into the vector space model. Section 3 derives a parametrised class of semantic proximity matrices from a recursive definition of similarity of terms and documents. A further parametrised class of kernels based on alternative similarity measures inspired by considering diffusion on a weighted graph of documents is given in Section 4. In Section 5 we show how the recently introduced alignment measure [2] can be used to perform model selection over the classes of kernels we have defined. Positive experimental results with the methods are reported in Section 5 before we draw conclusions in Section 6.

## 2 Representing Semantic Proximity

Kernel based methods are an attractive choice for inferring relations from textual data since they enable us to work in a document-by-document setting rather than in a term-by-term one [6]. In the vector space model, a document is represented by a vector indexed by the terms of the corpus. Hence, the vector will typically be sparse with non-zero entries for those terms occurring in the document. Two documents that use semantically related but distinct words will therefore show no similarity. The aim of a semantic proximity matrix [3] is to correct for this by indicating the strength of the relationship between terms that even though distinct are semantically related.

The semantic proximity matrix $P$ is indexed by pairs of terms $a$ and $b$, with the entry $P_{ab} = P_{ba}$ giving the strength of their semantic similarity. If the vectors corresponding to two documents are $\mathbf{d}_i$, $\mathbf{d}_j$, their inner product is now evaluated through the kernel

$$k(\mathbf{d}_i, \mathbf{d}_j) = \mathbf{d}_i' P \mathbf{d}_j,$$

where $\mathbf{x}'$ denotes the transpose of the vector or matrix $\mathbf{x}$. The symmetry of $P$ ensures that the kernel is symmetric. We must also require that $P$ is positive semi-definite in order to satisfy Mercer's conditions. In this case we can decompose $P = R'R$ for some matrix $R$, so that we can view the semantic similarity as a projection into a semantic space

$$\phi : \mathbf{d} \longmapsto R\mathbf{d}, \quad \text{since} \quad k(\mathbf{d}_i, \mathbf{d}_j) = \mathbf{d}_i' P \mathbf{d}_j = \langle R\mathbf{d}_i, R\mathbf{d}_j \rangle.$$

The purpose of this paper is to infer (or refine) the similarity measure between examples by taking into account higher order correlations, thereby performing unsupervised learning of the proximity matrix from a given corpus. We will propose two methods based on two different observations.

The first method exploits the fact that the standard representation of text documents as bags of words gives rise to an interesting duality: while documents can be seen as bags of words, simultaneously terms can be viewed as bags of documents – the documents that contain them. In such a model, two documents that have highly correlated term-vectors are considered as having similar content. Similarly, two terms that have a correlated document-vector will have a semantic relation. This is of course only a first order approximation since the knock-on effect of the two similarities on each other needs to be considered. We show that it is possible to define term-similarity based on document-similarity, and vice versa, to obtain a system of equations that can be solved in order to obtain a semantic proximity matrix $P$.

The second method exploits the representation of a lexicon (the set of all words in a given corpus) as a graph, where the nodes are indexed by words and where co-occurrence is used to establish links between nodes. Such a representation has been studied recently giving rise to a number of topological properties [4]. We consider

the idea that higher order correlations between terms can affect their semantic relations as a diffusion process on such a graph. Although there can be exponentially many paths connecting two given nodes in the graph, the use of diffusion kernels [7] enables us to obtain the level of semantic relation between any two nodes efficiently, so inferring the semantic proximity matrix from data.

## 3 Equilibrium Equations for Semantic Similarity

In this section we consider the first of the two methods outlined in the previous section. Here the aim is to create recursive equations for the relations between documents and between terms.

Let $X$ be the feature example (term/document in the case of text data) matrix in a possibly kernel-defined feature space, so that $X'X$ gives the kernel matrix $K$ and $XX'$ gives the correlations between different features over the training set. We denote this latter matrix with $G$. Consider the similarity matrices defined recursively by

$$\hat{K} = \lambda X'\hat{G}X + K \quad \text{and} \quad \hat{G} = \lambda X'\hat{K}X + G \tag{1}$$

We can interpret this as augmenting the similarity given by $K$ through indirect similarities measured by $G$ and vice versa. The factor $\lambda < \|K\|^{-1}$ ensures that the longer range effects decay exponentially. Our first result characterizes the solution of the above recurrences.

**Proposition 1** *Provided $\lambda < \|K\|^{-1} = \|G\|^{-1}$, the kernels $\hat{K}$ and $\hat{G}$ that solve the recurrences (1) are given by*

$$\hat{K} = K(I - \lambda K)^{-1} \quad \text{and} \quad \hat{G} = G(I - \lambda G)^{-1}$$

**Proof**: First observe that

$$
\begin{aligned}
K(I - \lambda K)^{-1} &= K(I - \lambda K)^{-1} - \frac{1}{\lambda}(I - \lambda K)^{-1} + \frac{1}{\lambda}(I - \lambda K)^{-1} \\
&= -\frac{1}{\lambda}(I - \lambda K)(I - \lambda K)^{-1} + \frac{1}{\lambda}(I - \lambda K)^{-1} \\
&= \frac{1}{\lambda}(I - \lambda K)^{-1} - \frac{1}{\lambda}I
\end{aligned}
$$

Now if we substitute the second recurrence into the first we obtain

$$
\begin{aligned}
\hat{K} &= \lambda^2 X'X\hat{K}X'X + \lambda X'XX'X + K \\
&= \lambda^2 K(K(I - \lambda K)^{-1})K + \lambda K^2 + K \\
&= \lambda^2 K(\frac{1}{\lambda}(I - \lambda K)^{-1} - \frac{1}{\lambda}I)K + \lambda K^2 + K \\
&= \lambda K(I - \lambda K)^{-1}K + K(I - \lambda K)^{-1}(I - \lambda K) \\
&= K(I - \lambda K)^{-1}
\end{aligned}
$$

showing that the expression does indeed satisfy the recurrence. Clearly, by the symmetry of the definition the expression for $\hat{G}$ also satisfies the recurrence. ∎

In view of the form of the solution we introduce the following definition:

**Definition 2 von Neumann Kernel** *Given a kernel $K$ the derived kernel $\hat{K}(\lambda) = K(I - \lambda K)^{-1}$ will be referred to as the von Neumann kernel.*

Note that we can view $\hat{K}(\lambda)$ as a kernel based on the semantic proximity matrix $P = \lambda\hat{G} + I$ since

$$X'PX = X'(\lambda\hat{G} + I)X = \lambda X'\hat{G}X + K = \hat{K}(\lambda).$$

Hence, the solution $\hat{G}$ defines a refined similarity between terms/features. In the next section, we will consider the second method of introducing semantic similarity derived from viewing the terms and documents as vertices of a weighted graph.

## 4  Semantic Similarity as a Diffusion Process

Graph like structures within data occur frequently in many diverse settings. In the case of language, the topological structure of a lexicon graph has recently been analyzed [4]. Such a graph has nodes indexed by all the terms in the corpus, and the edges are given by the co-occurrence between terms in documents of the corpus. Although terms that are connected are likely to have related meaning, terms with a higher degree of separation would not be considered as being related.

A diffusion process on the graph can also be considered as a model of semantic relations existing between indirectly connected terms. Although the number of possible paths between any two given nodes can grow exponentially, results from spectral graph theory have been recently used by [7] to show that it is possible to compute the similarity between any two given nodes efficiently *without* examining all possible paths. It is also possible to show that the similarity measure obtained in this way is a valid kernel function. The exponentiation operation used in the definition, naturally yields the Mercer conditions required for valid kernel functions.

An alternative insight into semantic similarity, to that presented in section 2, is afforded if we multiply out the expression for $\hat{K}(\lambda)$, $\hat{K}(\lambda) = K(I - \lambda K)^{-1} = \sum_{t=1}^{\infty} \lambda^{t-1}K^t$. The entries in the matrix $K^t$ are given by

$$K_{ij}^t = \sum_{\substack{u \in \{1,\ldots,m\}^t \\ u_1 = i, u_t = j}} \prod_{\ell=1}^{t-1} K_{u_\ell u_{\ell+1}},$$

that is the sum of the products of the weights over all paths of length $t$ that start at vertex $i$ and finish at vertex $j$ in the weighted graph on the examples. If we view the connection strengths as channel capacities, the entry $K_{ij}^t$ can be seen to measure the sum over all routes of the products of the capacities. If the entries satisfy that they are all positive and for each vertex the sum of the connections is 1, we can view the entry as the probability that a random walk beginning at vertex $i$ is at vertex $j$ after $t$ steps. It is for these reasons that the kernels defined using these combinations of powers of the kernel matrix have been termed diffusion kernels [7]. A similar equation holds for $G^t$. Hence, examples that both lie in a cluster of similar examples become more strongly related, and similar features that occur in a cluster of related features are drawn together in the semantic proximity matrix $P$. We should stress that the emphasis of this work is not in its diffusion connections, but its relation to semantic proximity. It is this link that motivates the alternative decay factors considered below.

The kernel $\hat{K}$ combines these indirect link kernels with an exponentially decaying weight. This suggests an alternative weighting scheme that shows faster decay for increasing path length,

$$\tilde{K}(\lambda) = K \sum_{t=1}^{\infty} \frac{\lambda^t K^t}{t!} = K \exp(\lambda K)$$

The next proposition gives the semantic proximity matrix corresponding to $\tilde{K}(\lambda)$.

**Proposition 3** *Let* $\tilde{K}(\lambda) = K \exp(\lambda K)$. *Then* $\tilde{K}(\lambda)$ *corresponds to a semantic proximity matrix* $\exp(\lambda G)$.

**Proof**: Let $X = U\Sigma V'$ be the singular value decomposition of $X$, so that $K = V\Lambda V'$ is the eigenvalue decomposition of $K$, where $\Lambda = \Sigma'\Sigma$. We can write $\tilde{K}$ as

$$
\begin{aligned}
\tilde{K} &= V\Lambda \exp(\lambda\Lambda)V' = X'U\Sigma^{-1}\Lambda\exp(\lambda\Lambda)\Sigma^{-1}U'X \\
&= X'U\exp(\lambda\Lambda)U'X = X'\exp(\lambda G)X, \quad \text{as required. } \blacksquare
\end{aligned}
$$

The above leads to the definition of the second kernel that we consider.

**Definition 4** *Given a kernel $K$ the derived kernels $\hat{K}(\lambda) = K \exp(\lambda K)$ will be referred to as the exponential kernels.*

## 5 Experimental Methods

In the previous sections we have introduced two new kernel adaptations, in both cases parameterized by a positive real parameter $\lambda$. In order to apply these kernels on real text data, we need to develop a method of choosing the parameter $\lambda$. Of course one possibility would be just to use cross-validation as considered by [7]. Rather than adopt this rather expensive methodology we will use a quantitative measure of agreement between the diffusion kernels and the learning task known as *alignment*, which measures the degree of agreement between a kernel and target [2].

**Definition 5 Alignment** *The (empirical) alignment of a kernel $k_1$ with a kernel $k_2$ with respect to the sample $S$ is the quantity*

$$
A(S, k_1, k_2) = \frac{\langle K_1, K_2\rangle_F}{\sqrt{\langle K_1, K_1\rangle_F \langle K_2, K_2\rangle_F}},
$$

*where $K_i$ is the kernel matrix for the sample $S$ using kernel $k_i$.*

where we use the following definition of inner products between Gram matrices

$$
\langle K_1, K_2\rangle_F = \sum_{i,j=1}^{m} K_1(x_i, x_j)K_2(x_i, x_j) \tag{2}
$$

corresponding to the Frobenius inner product. From a text categorization perspective this can also be viewed as the cosine of the angle between two bi-dimensional vectors $K_1$ and $K_2$, representing the Gram matrices. If we consider $K_2 = yy'$, where $y$ is the vector of outputs (+1/-1) for the sample, then

$$
A(S, K, yy') = \frac{\langle K, yy'\rangle_F}{\sqrt{\langle K, K\rangle_F \langle yy', yy'\rangle_F}} = \frac{y'Ky}{m\|K\|_F} \tag{3}
$$

The alignment has been shown to possess several convenient properties [2]. Most notably it can be efficiently computed before any training of the kernel machine takes place, and based only on training data information; and since it is sharply concentrated around its expected value, its empirical value is stable with respect to different splits of the data.

We have developed a method for choosing $\lambda$ to optimize the alignment of the resulting matrix $\tilde{K}(\lambda)$ or $\hat{K}(\lambda)$ to the target labels on the training set. This method

follows similar results presented in [2], but here the parameterization is non-linear in $\lambda$ so that we cannot solve for the optimal value. We rather seek the optimal value using a line search over the range of possible values of $\lambda$ for the value at which the derivative of the alignment with respect to $\lambda$ is zero. The next two propositions will give equations that are satisfied at this point.

**Proposition 6** *If $\lambda^\star$ is the solution of $\lambda^\star = \mathrm{argmax}_\lambda A(S, \tilde{K}(\lambda), yy')$ and $v_i, \lambda_i$ are the eigenvector/eigenvalue pairs of the kernel matrix $K$ then*

$$\sum_{i=1}^m \lambda_i^2 \exp(2\lambda^\star \lambda_i) \sum_{i=1}^m \langle v_i, y \rangle^2 \exp(\lambda^\star \lambda_i)\lambda_i^2 \;=\; \sum_{i=1}^m \lambda_i \exp(\lambda^\star \lambda_i)\langle v_i, y \rangle^2 \sum_{i=1}^m \lambda_i^3 \exp(2\lambda^\star \lambda_i)$$

**Proof**: First observe that $\tilde{K}(\lambda) = VMV' = \sum_{i=1}^m \mu_i v_i v_i'$, where $M_{ii} = \mu_i(\lambda) = \lambda_i \exp(\lambda \lambda_i)$. We can express the alignment of $\tilde{K}(\lambda)$ as

$$A(S, \tilde{K}(\lambda), yy') \;=\; \frac{\sum_{i=1}^m \mu_i(\lambda)\langle v_i, y \rangle^2}{m\sqrt{\sum_{i=1}^m \mu_i(\lambda)^2}}.$$

The function is a differentiable function of $\lambda$ and so at its maximal value the derivative will be zero. Taking the derivative of this expression and setting it equal to zero gives the condition in the proposition statement. ∎

**Proposition 7** *If $\lambda^\star$ is the solution of $\lambda^\star = \mathrm{argmax}_{\lambda \in (0, \|K\|^{-1})} A(S, \tilde{K}(\lambda), yy')$, and $v_i, \lambda_i$ are the eigenvector eigenvalue pairs of the kernel matrix $K$ then*

$$\sum_{i=1}^m \frac{1}{(\lambda^\star(1 - \lambda^\star \lambda_i))^2} \sum_{i=1}^m \frac{\langle v_i, y \rangle^2 (2\lambda^\star \lambda_i - 1)}{(\lambda^\star(1 - \lambda^\star \lambda_i))^2} \;=\; \sum_{i=1}^m \frac{\langle v_i, y \rangle^2}{\lambda^\star(1 - \lambda^\star \lambda_i)} \sum_{i=1}^m \frac{2\lambda^\star \lambda_i - 1}{(\lambda^\star(1 - \lambda^\star \lambda_i))^3}$$

**Proof**: The proof is identical to that of Proposition 6 except that $M_{ii} = \mu_i(\lambda) = \frac{(1 - \lambda_i \lambda)^{-1}}{\lambda}$. ∎

**Definition 8 Line Search** *Optimization of the alignment can take place by using a line search of the values of $\lambda$ to find a maximum point of the alignment by seeking points at which the equations given in Proposition 6 and 7 hold.*

## 5.1 Results

To demonstrate the performance of the proposed algorithm for text data, the Medline1033 dataset commonly used in text processing [3] was used. This dataset contains 1033 documents and 30 queries obtained from the national library of medicine. In this work we focus on query20. A Bag of Words kernel was used [6]. Stop words and punctuation were removed from the documents and the Porter stemmer was applied to the words. The terms in the documents were weighted according to a variant of the $tfidf$ scheme. It is given by $\log(1 + tf) * \log(m/df)$, where $tf$ represents the term frequency, $df$ is used for the document frequency and $m$ is the total number of documents. A support vector classifier (SVC) was used to assess the performance of the derived kernels on the Medline dataset. A 10-fold cross validation procedure was used to find the optimal value for the capacity control parameter 'C'. Having selected the optimal 'C' parameter, the SVC was re-trained ten times using ten random training and test dataset splits. Error results for the different algorithms are presented together with $F1$ values. The $F1$ measure is a popular statistic used in the information retrieval community for comparing performance of

|  | Train Align | SVC Error | F1 | $\lambda$ |
|---|---|---|---|---|
| $K_{80}$ | **0.851 (0.012)** | **0.017 (0.005)** | **0.795 (0.060)** | 0.197 (0.004) |
| $B_{80}$ | 0.423 (0.007) | 0.022 (0.007) | 0.256 (0.351) | - |
| $K_{50}$ | **0.863 (0.025)** | **0.018 (0.006)** | **0.783 (0.074)** | 0.185 (0.008) |
| $B_{50}$ | 0.390 (0.009) | 0.024 (0.004) | 0.456 (0.265) | - |
| $K_{20}$ | **0.867 (0.029)** | **0.019 (0.004)** | **0.731 (0.089)** | 0.147 (0.04) |
| $B_{20}$ | 0.325 (0.009) | 0.030 (0.005) | 0.349 (0.209) | - |

Table 1: Medline dataset - Mean and associated standard deviation alignment, $F1$ and SVC error values for a SVC trained using the Bag of Words kernel (B) and the exponential kernel (K). The index represents the percentage of training points.

|  | Train Align | SVC Error | F1 | $\lambda$ |
|---|---|---|---|---|
| $\hat{K}_{80}$ | **0.758 (0.015)** | **0.017 (0.004)** | **0.765 (0.020)** | 0.032 (0.001) |
| $B_{80}$ | 0.423(0.007) | 0.022 (0.007) | 0.256 (0.351) | - |
| $\hat{K}_{50}$ | **0.766 (0.025)** | **0.018 (0.005)** | **0.701 (0.066)** | 0.039 (0.008) |
| $B_{50}$ | 0.390 (0.009) | 0.024 (0.004) | 0.456 (0.265) | - |
| $\hat{K}_{20}$ | **0.728 (0.012)** | **0.028 (0.004)** | **0.376 (0.089)** | 0.029 (0.07) |
| $B_{20}$ | 0.325 (0.009) | 0.030 (0.005) | 0.349 (0.209) | - |

Table 2: Medline dataset - Mean and associated standard deviation alignment, $F1$ and SVC error values for a SVC trained using the Bag of Words kernel (B) and the von Neumann ($\hat{K}$). The index represents the percentage of training points.

algorithms typically on uneven data. $F1$ can be computed using $F1 = \frac{2\tilde{P}R}{\tilde{P}+R}$, where $\tilde{P}$ represents precision i.e. a measure of the proportion of selected items that the system classified correctly, and R represents recall i.e. the proportion of the target items that the system selected.

Applying the line search procedure to find the optimal value of $\lambda$ for the diffusion kernels. All of the results are averaged over 10 random splits with the standard deviation given in brackets. Table 1 shows the results of using the Bag of Words kernel matrix (B) and the exponential kernel matrix (K). Table 2 presents the results of using the von Neumann kernel matrix ($\hat{K}$) together with the Bag of Words kernel matrix for different sizes of the training data. The index represents the percentage of training points. The first column of both table 1 and 2 shows the alignments of the Gram matrices to the rank 1 labels matrix for different sizes of training data.

In both cases the results presented indicate that the alignment of the diffusion kernels to the labels is greater than that of the Bag of Words kernel matrix by more than the sum of the standard deviations across all sizes of training data. The second column of the tables represents the support vector classifier (SVC) error obtained using the diffusion Gram matrices and the Bag of Words Gram matrix. The SVC error for the diffusion kernels shows a decrease with increasing alignment value. $F1$ values are also shown and in all instances show an improvement for the diffusion kernel matrices. An interesting observation can be made regarding the $F1$ value for the von Neumann kernel matrix trained using 20% training data ($\hat{K}_{20}$). Despite an increase in alignment value and reduction of SVC error the $F1$ value does not increase as much as that for the exponential kernel trained using the same proportion of the data $K_{20}$. This observation implies that the diffusion kernel needs

more data to be effective. This will be investigated in future work.

## 6 Conclusions

We have proposed and compared two different methods to model the notion of semantic similarity between documents, by implicitly defining a proximity matrix $P$ in a way that exploits high order correlations between terms. The two methods differ in the way the matrix is constructed. In one view, we propose a recursive definition of document similarity that depends on term similarity and vice versa. By solving the resulting system of kernel equations, we effectively learn the parameters of the model ($P$), and construct a kernel function for use in kernel based learning methods. In the other approach, we model semantic relations as a diffusion process in a graph whose nodes are the documents and edges incorporate first-order similarity. Diffusion efficiently takes into account all possible paths connecting two nodes, and propagates the 'similarity' between two remote documents that share 'similar terms'. The kernel resulting from this model is known in the literature as the 'diffusion kernel'. We have experimentally demonstrated the validity of the approach on text data using a novel approach to set the adjustable parameter $\lambda$ in the kernels by optimising their 'alignment' to the target on the training set. For the dataset partitions substantial improvements in performance over the traditional Bag of Words kernel matrix were obtained using the diffusion kernels and the line search method. Despite this success, for large imbalanced datasets such as those encountered in text classification tasks the computational complexity of constructing the diffusion kernels may become prohibitive. Faster kernel construction methods are being investigated for this regime.

## References

[1] N. Cristianini and J. Shawe-Taylor. *An Introduction to Support Vector Machines*. Cambridge University Press, Cambridge, UK, 2000.

[2] Nello Cristianini, John Shawe-Taylor, and Jaz Kandola. On kernel target alignment. In *Proceedings of the Neural Information Processing Systems, NIPS'01*, 2002.

[3] Nello Cristianini, John Shawe-Taylor, and Huma Lodhi. Latent semantic kernels. *Journal of Intelligent Information Systems*, 18(2):127–152, 2002.

[4] R. Ferrer and R.V. Sole. The small world of human language. *Proceedings of the Royal Society of London Series B - Biological Sciences*, pages 2261–2265, 2001.

[5] Thomas Hofmann. Probabilistic latent semantic indexing. In *Research and Development in Information Retrieval*, pages 50–57, 1999.

[6] T. Joachims. Text categorization with support vector machines. In *Proceedings of European Conference on Machine Learning (ECML)*, 1998.

[7] R.I. Kondor and J. Lafferty. Diffusion kernels on graphs and other discrete structures. In *Proceedings of Intenational Conference on Machine Learning (ICML 2002)*, 2002.

[8] Todd A. Letsche and Michael W. Berry. Large-scale information retrieval with latent semantic indexing. *Information Sciences*, 100(1-4):105–137, 1997.

[9] G. Siolas and F. d'Alch Buc. Support vector machines based on a semantic kernel for text categorization. In *IEEE-IJCNN 2000)*, 2000.
